# Co-Regularized Hashing for Multimodal Data

**Yi Zhen and Dit-Yan Yeung**
Department of Computer Science and Engineering
Hong Kong University of Science and Technology
Clear Water Bay, Kowloon, Hong Kong
{yzhen,dyyeung}@cse.ust.hk

## Abstract

Hashing-based methods provide a very promising approach to large-scale similarity search. To obtain compact hash codes, a recent trend seeks to learn the hash functions from data automatically. In this paper, we study hash function learning in the context of multimodal data. We propose a novel multimodal hash function learning method, called Co-Regularized Hashing (CRH), based on a boosted co-regularization framework. The hash functions for each bit of the hash codes are learned by solving DC (difference of convex functions) programs, while the learning for multiple bits proceeds via a boosting procedure so that the bias introduced by the hash functions can be sequentially minimized. We empirically compare CRH with two state-of-the-art multimodal hash function learning methods on two publicly available data sets.

## 1 Introduction

Nearest neighbor search, *a.k.a.* similarity search, plays a fundamental role in many important applications, including document retrieval, object recognition, and near-duplicate detection. Among the methods proposed thus far for nearest neighbor search [1], hashing-based methods [2, 3] have attracted considerable interest in recent years. The major advantage of hashing-based methods is that they index data using binary hash codes which enjoy not only low storage requirements but also high computational efficiency. To preserve similarity in the data, a family of algorithms called locality sensitive hashing (LSH) [4, 5] has been developed over the past decade. The basic idea of LSH is to hash the data into bins so that the collision probability reflects data similarity. LSH is very appealing in that it has theoretical guarantee and is also simple to implement. However, in practice LSH algorithms often generate long hash codes in order to achieve acceptable performance because the theoretical guarantee only holds asymptotically. This shortcoming can be attributed largely to their data-independent nature which cannot capture the data characteristics very accurately in the hash codes. Besides, in many applications, neighbors cannot be defined easily using some generic distance or similarity measures. As such, a new research trend has emerged over the past few years by learning the hash functions from data automatically. In the sequel, we refer to this new trend as hash function learning (HFL).

Boosting, as one of the most popular machine learning approaches, was first applied to learning hash functions for pose estimation [6]. Later, impressive performance for HFL using restricted Boltzmann machines was reported [7]. These two early HFL methods have been successfully applied to content-based image retrieval in which large-scale data sets are commonly encountered [8]. A number of algorithms have been proposed since then. Spectral hashing (SH) [9] treats HFL as a special case of manifold learning and uses an efficient algorithm based on eigenfunctions. One shortcoming of spectral hashing is in its assumption, which requires that the data be uniformly distributed. To overcome this limitation, several methods have been proposed, including binary reconstructive embeddings [10], shift-invariant kernel hashing [11], distribution matching [12], optmized kernel hashing [13], and minimal loss hashing [14]. Recently, some semi-supervised hashing models have

been developed to combine both feature similarity and semantic similarity for HFL [15, 16, 17, 18]. To further improve the scalability of these methods, Liu *et al.* [19] presented a fast algorithm based on anchor graphs.

Existing HFL algorithms have enjoyed wide success in challenging applications. Nevertheless, they can only be applied to a single type of data, called unimodal data, which refer to data from a single modality such as image, text, or audio. Nowadays, it is common to find similarity search applications that involve multimodal data. For example, given an image of a tourist attraction as query, one would like to retrieve some textual documents that provide more detailed information about the place of interest. Because data from different modalities reside in different feature spaces, performing multimodal similarity search will be made much easier and faster if the multimodal data can be mapped into a common Hamming space. However, it is challenging to do so because data from different modalities generally have very different representations.

As far as we know, there exist only two multimodal HFL methods. Bronstein *et al.* [20] made the first attempt to learn linear hash functions using eigendecomposition and boosting, while Kumar *et al.* [21] extended spectral hashing to the multiview setting and proposed a cross-view hashing model. One major limitation of these two methods is that they both rely on eigendecomposition operations which are computationally very demanding when the data dimensionality is high. Moreover, they consider applications for shape retrieval, image alignment, and people search which are quite different from the multimodal retrieval applications of interest here.

In this paper, we propose a novel multimodal HFL method, called Co-Regularized Hashing (CRH), based on a boosted co-regularization framework. For each bit of the hash codes, CRH learns a group of hash functions, one for each modality, by minimizing a novel loss function. Although the loss function is non-convex, it is in a special form which can be expressed as a difference of convex functions. As a consequence, the Concave-Convex Procedure (CCCP) [22] can be applied to solve the optimization problem iteratively. We use a stochastic sub-gradient method, which converges very fast, in each CCCP iteration to find a local optimum. After learning the hash functions for one bit, CRH proceeds to learn more bits via a boosting procedure such that the bias introduced by the hash functions can be sequentially minimized.

In the next section, we present the CRH method in detail. Extensive empirical study using two data sets is reported in Section 3. Finally, Section 4 concludes the paper.

## 2   Co-Regularized Hashing

We use boldface lowercase letters and calligraphic letters to denote vectors and sets, respectively. For a vector $\mathbf{x}$, $\mathbf{x}^T$ denotes its transpose and $\|\mathbf{x}\|$ its $\ell_2$ norm.

### 2.1   Objective Function

Suppose that there are two sets of data points from two modalities,[1] e.g., $\{\mathbf{x}_i \in \mathcal{X}\}_{i=1}^I$ for a set of $I$ images from some feature space $\mathcal{X}$ and $\{\mathbf{y}_j \in \mathcal{Y}\}_{j=1}^J$ for a set of $J$ textual documents from another feature space $\mathcal{Y}$. We also have a set of $N$ inter-modality point pairs $\Theta = \{(\mathbf{x}_{a_1}, \mathbf{y}_{b_1}), (\mathbf{x}_{a_2}, \mathbf{y}_{b_2}), \ldots, (\mathbf{x}_{a_N}, \mathbf{y}_{b_N})\}$, where, for the $n$th pair, $a_n$ and $b_n$ are indices of the points in $\mathcal{X}$ and $\mathcal{Y}$, respectively. We further assume that each pair has a label $s_n = 1$ if $\mathbf{x}_{a_n}$ and $\mathbf{y}_{b_n}$ are similar and $s_n = 0$ otherwise. The notion of inter-modality similarity varies from application to application. For example, if an image includes a tiger and a textual document is a research paper on tigers, they should be labeled as similar. On the other hand, it is highly unlikely to label the image as similar to a textual document on basketball.

For each bit of the hash codes, we define two linear hash functions as follows:

$$f(\mathbf{x}) = \mathrm{sgn}(\mathbf{w}_x^T \mathbf{x}) \ \text{ and } \ g(\mathbf{y}) = \mathrm{sgn}(\mathbf{w}_y^T \mathbf{y}),$$

where $\mathrm{sgn}(\cdot)$ denotes the sign function, and $\mathbf{w}_x$ and $\mathbf{w}_y$ are projection vectors which, ideally, should map similar points to the same hash bin and dissimilar points to different bins. Our goal is to achieve HFL by learning $\mathbf{w}_x$ and $\mathbf{w}_y$ from the multimodal data.

To achieve this goal, we propose to minimize the following objective function *w.r.t.* (with respect to) $\mathbf{w}_x$ and $\mathbf{w}_y$:

$$\mathcal{O} = \frac{1}{I}\sum_{i=1}^{I}\ell_i^x + \frac{1}{J}\sum_{j=1}^{J}\ell_j^y + \gamma\sum_{n=1}^{N}\omega_n\ell_n^* + \frac{\lambda_x}{2}\|\mathbf{w}_x\|^2 + \frac{\lambda_y}{2}\|\mathbf{w}_y\|^2, \tag{1}$$

where $\ell_i^x$ and $\ell_j^y$ are intra-modality loss terms for modalities $\mathcal{X}$ and $\mathcal{Y}$, respectively. In this work, we define them as:

$$\ell_i^x = \left[1 - f(\mathbf{x}_i)(\mathbf{w}_x^T\mathbf{x}_i)\right]_+ = \left[1 - |\mathbf{w}_x^T\mathbf{x}_i|\right]_+,$$
$$\ell_j^y = \left[1 - g(\mathbf{y}_j)(\mathbf{w}_y^T\mathbf{y}_j)\right]_+ = \left[1 - |\mathbf{w}_y^T\mathbf{y}_j|\right]_+,$$

where $[a]_+$ is equal to $a$ if $a \geq 0$ and 0 otherwise. We note that the intra-modality loss terms are similar to the hinge loss in the (linear) support vector machine but have quite different meaning. Conceptually, we want the projected values to be far away from 0 and hence expect the hash functions learned to have good generalization ability [16]. For the inter-modality loss term $\ell_n^*$, we associate with each point pair a weight $\omega_n$, with $\sum_{n=1}^{N}\omega_n = 1$, to normalize the loss as well as compute the bias of the hash functions. In this paper, we define $\ell_n^*$ as

$$\ell_n^* = s_n d_n^2 + (1 - s_n)\tau(d_n),$$

where $d_n = \mathbf{w}_x^T\mathbf{x}_{a_n} - \mathbf{w}_y^T\mathbf{y}_{b_n}$ and $\tau(d)$ is called the smoothly clipped inverted squared deviation (SCISD) function. The loss function such defined requires that the similar inter-modality points, i.e., $s_n = 1$, have small distance after projection, and the dissimilar ones, i.e., $s_n = 0$, have large distance. With these two kinds of loss terms, we expect that the learned hash functions can enjoy the large-margin property while effectively preserving the inter-modality similarity.

The SCISD function was first proposed in [23]. It can be defined as follows:

$$\tau(d) = \begin{cases} -\frac{1}{2}d^2 + \frac{a\lambda^2}{2} & \text{if } |d| \leq \lambda \\ \frac{d^2 - 2a\lambda|d| + a^2\lambda^2}{2(a-1)} & \text{if } \lambda < |d| \leq a\lambda \\ 0 & \text{if } a\lambda < |d|, \end{cases}$$

where $a$ and $\lambda$ are two user-specified parameters. The SCISD function penalizes projection vectors that result in small distance between dissimilar points after projection. A more important property is that it can be expressed as a difference of two convex functions. Specifically, we can express $\tau(d) = \tau_1(d) - \tau_2(d)$ where

$$\tau_1(d) = \begin{cases} 0 & \text{if } |d| \leq \lambda \\ \frac{ad^2 - 2a\lambda|d| + a\lambda^2}{2(a-1)} & \text{if } \lambda < |d| \leq a\lambda \\ \frac{1}{2}d^2 - \frac{a\lambda^2}{2} & \text{if } a\lambda < |d| \end{cases} \quad \text{and} \quad \tau_2(d) = \frac{1}{2}d^2 - \frac{a\lambda^2}{2}.$$

## 2.2  Optimization

Though the objective function (1) is nonconvex *w.r.t.* $\mathbf{w}_x$ and $\mathbf{w}_y$, we can optimize it *w.r.t.* $\mathbf{w}_x$ and $\mathbf{w}_y$ in an alternating manner. Take $\mathbf{w}_x$ for example, we remove the irrelevant terms and get the following objective:

$$\frac{1}{I}\sum_{i=1}^{I}\ell_i^x + \frac{\lambda_x}{2}\|\mathbf{w}_x\|^2 + \gamma\sum_{n=1}^{N}\omega_n\ell_n^*, \tag{2}$$

where

$$\ell_i^x = \begin{cases} 0 & \text{if } |\mathbf{w}_x^T\mathbf{x}_i| \geq 1 \\ 1 - \mathbf{w}_x^T\mathbf{x}_i & \text{if } 0 \leq \mathbf{w}_x^T\mathbf{x}_i < 1 \\ 1 + \mathbf{w}_x^T\mathbf{x}_i & \text{if } -1 < \mathbf{w}_x^T\mathbf{x}_i < 0. \end{cases}$$

It is easy to realize that the objective function (2) can be expressed as a difference of two convex functions in different cases. As a consequence, we can use CCCP to solve the nonconvex optimization problem iteratively with each iteration minimizing a convex upper bound of the original objective function.

Briefly speaking, given an objective function $f_0(x) - g_0(x)$ where both $f_0$ and $g_0$ are convex, CCCP works iteratively as follows. The variable $x$ is first randomly initialized to $x^{(0)}$. At the $t$th iteration, CCCP minimizes the following convex upper bound of $f_0(x) - g_0(x)$ at location $x^{(t)}$:

$$f_0(x) - \left( g_0(x^{(t)}) + \partial_x g_0(x^{(t)})(x - x^{(t)}) \right),$$

where $\partial_x g_0(x^{(t)})$ is the first derivative of $g_0(x)$ at $x^{(t)}$. This optimization problem can be solved using any convex optimization solver to obtain $x^{(t+1)}$. Given an initial value $x^{(0)}$, the solution sequence $\{x^{(t)}\}$ found by CCCP is guaranteed to reach a local minimum or a saddle point.

For our problem, the optimization problem at the $t$th iteration minimizes the following upper bound of Equation (2) w.r.t. $\mathbf{w}_x$:

$$\mathcal{O}_x = \frac{\lambda_x \|\mathbf{w}_x\|^2}{2} + \gamma \sum_{n=1}^{N} \omega_n \left( s_n d_n^2 + (1 - s_n)\zeta_n^x \right) + \frac{1}{I} \sum_{i=1}^{I} \ell_i^x, \tag{3}$$

where $\zeta_n^x = \tau_1(d_n) - \tau_2(d_n^{(t)}) - d_n^{(t)} \mathbf{x}_{a_n}^T (\mathbf{w}_x - \mathbf{w}_x^{(t)})$, $d_n^{(t)} = (\mathbf{w}_x^{(t)})^T \mathbf{x}_{a_n} - \mathbf{w}_y^T \mathbf{y}_{b_n}$, and $\mathbf{w}_x^{(t)}$ is the value of $\mathbf{w}_x$ at the $t$th iteration.

To find a locally optimal solution to problem (3), we can use any gradient-based method. In this work, we develop a stochastic sub-gradient solver based on Pegasos [24], which is known to be one of the fastest solvers for margin-based classifiers. Specifically, we randomly select $k$ points from each modality and $l$ point pairs to evaluate the sub-gradient at each iteration.

The key step of our method is to evaluate the sub-gradient of objective function (3) w.r.t. $\mathbf{w}_x$, which can be computed as

$$\frac{\partial \mathcal{O}_x}{\partial \mathbf{w}_x} = 2\gamma \sum_{n=1}^{N} \omega_n s_n d_n \mathbf{x}_{a_n} + \gamma \sum_{n=1}^{N} \omega_n \boldsymbol{\mu}_n^x + \lambda_x \mathbf{w}_x - \frac{1}{I} \sum_{i=1}^{I} \boldsymbol{\pi}_i^x, \tag{4}$$

where $\boldsymbol{\mu}_n^x = (1 - s_n) \left( \frac{\partial \tau_1}{\partial d_n} - d_n^{(t)} \right) \mathbf{x}_{a_n}$,

$$\frac{\partial \tau_1}{\partial d_n} = \begin{cases} 0 & \text{if } |d_n| \le \lambda \\ \frac{ad_n - 2a\lambda \operatorname{sgn}(d_n)}{(a-1)} & \text{if } \lambda < |d_n| \le a\lambda \\ d_n & \text{if } a\lambda < |d_n| \end{cases} \quad \text{and} \quad \boldsymbol{\pi}_i^x = \begin{cases} 0 & \text{if } |\mathbf{w}_x^T \mathbf{x}_i| \ge 1 \\ \operatorname{sgn}\left(\mathbf{w}_x^T \mathbf{x}_i\right) \mathbf{x}_i & \text{if } |\mathbf{w}_x^T \mathbf{x}_i| < 1. \end{cases}$$

Similarly, the objective function for the optimization problem w.r.t. $\mathbf{w}_y$ at the $t$th CCCP iteration is:

$$\mathcal{O}_y = \frac{\lambda_y \|\mathbf{w}_y\|^2}{2} + \gamma \sum_{n=1}^{N} \omega_n \left( s_n d_n^2 + (1 - s_n)\zeta_n^y \right) + \frac{1}{J} \sum_{j=1}^{I} \ell_j^y, \tag{5}$$

where $\zeta_n^y = \tau_1(d_n) - \tau_2(d_n^{(t)}) + d_n^{(t)} \mathbf{y}_{b_n}^T (\mathbf{w}_y - \mathbf{w}_y^{(t)})$, $d_n^{(t)} = \mathbf{w}_x^T \mathbf{x}_{a_n} - (\mathbf{w}_y^{(t)})^T \mathbf{y}_{b_n}$, $\mathbf{w}_y^{(t)}$ is the value of $\mathbf{w}_y$ at the $t$th iteration and

$$\ell_j^y = \begin{cases} 0 & \text{if } |\mathbf{w}_y^T \mathbf{y}_j| \ge 1 \\ 1 - \mathbf{w}_y^T \mathbf{y}_j & \text{if } 0 \le \mathbf{w}_y^T \mathbf{y}_j < 1 \\ 1 + \mathbf{w}_y^T \mathbf{y}_j & \text{if } -1 < \mathbf{w}_y^T \mathbf{y}_j < 0. \end{cases}$$

The corresponding sub-gradient is given by

$$\frac{\partial \mathcal{O}_y}{\partial \mathbf{w}_y} = -2\gamma \sum_{n=1}^{N} \omega_n s_n d_n \mathbf{y}_{b_n} - \gamma \sum_{n=1}^{N} \omega_n \boldsymbol{\mu}_n^y + \lambda_y \mathbf{w}_y - \frac{1}{J} \sum_{j=1}^{I} \boldsymbol{\pi}_j^y, \tag{6}$$

where $\boldsymbol{\mu}_n^y = (1 - s_n) \left( \frac{\partial \tau_1}{\partial d_n} - d_n^{(t)} \right) \mathbf{y}_{b_n}$ and

$$\boldsymbol{\pi}_j^y = \begin{cases} 0 & \text{if } |\mathbf{w}_y^T \mathbf{y}_j| \ge 1 \\ \operatorname{sgn}\left(\mathbf{w}_y^T \mathbf{y}_j\right) \mathbf{y}_j & \text{if } |\mathbf{w}_y^T \mathbf{y}_j| < 1. \end{cases}$$

## 2.3 Algorithm

So far we have only discussed how to learn the hash functions for one bit of the hash codes. To learn the hash functions for multiple bits, one could repeat the same procedure and treat the learning for each bit independently. However, as reported in previous studies [15, 19], it is very important to take into consideration the relationships between different bits in HFL. In other words, to learn compact hash codes, we should coordinate the learning of hash functions for different bits.

To this end, we take the standard AdaBoost [25] approach to learn multiple bits sequentially. Intuitively, this approach allows learning of the hash functions in later stages to be aware of the bias introduced by their antecedents. The overall algorithm of CRH is summarized in Algorithm 1.

---

**Algorithm 1** Co-Regularized Hashing

**Input:**
$\mathcal{X}, \mathcal{Y}$ – multimodal data
$\Theta$ – inter-modality point pairs
$K$ – code length
$\lambda_x, \lambda_y, \gamma$ – regularization parameters
$a, \lambda$ – parameters for SCISD function
**Output:**
$\mathbf{w}_x^{(k)}, k = 1, \ldots, K$ – projection vectors for $\mathcal{X}$
$\mathbf{w}_y^{(k)}, k = 1, \ldots, K$ – projection vectors for $\mathcal{Y}$

**Procedure:**
Initialize $\omega_n^{(1)} = 1/N, \forall n \in \{1, 2, \ldots, N\}$.
**for** $k = 1$ **to** $K$ **do**
  **repeat**
    Optimize Equation (3) to get $\mathbf{w}_x^{(k)}$;
    Optimize Equation (5) to get $\mathbf{w}_y^{(k)}$;

**until** convergence.
Compute error of current hash functions

$$\epsilon_k = \sum_{n=1}^{N} \omega_n^{(k)} \mathbf{I}_{[s_n \neq h_n]},$$

where $\mathbf{I}_{[a]} = 1$ if $a$ is true and $\mathbf{I}_{[a]} = 0$ otherwise, and

$$h_n = \begin{cases} 1 & \text{if } f(\mathbf{x}_{a_n}) = g(\mathbf{y}_{b_n}) \\ 0 & \text{if } f(\mathbf{x}_{a_n}) \neq g(\mathbf{y}_{b_n}). \end{cases}$$

Set $\beta_k = \epsilon_k / (1 - \epsilon_k)$.
Update the weight for each point pair:

$$\omega_n^{(k+1)} = \omega_n^{(k)} \beta_k^{1 - \mathbf{I}_{[s_n \neq h_n]}}.$$

**end for**

---

In the following, we briefly analyze the time complexity of Algorithm 1 for one bit. The first computationally expensive part of the algorithm is to evaluate the sub-gradients. The time complexity is $O((k + l)d)$, where $d$ is the data dimensionality, and $k$ and $l$ are the numbers of random points and random pairs, respectively, for the stochastic sub-gradient solver. In our experiments, we set $k = 1$ and $l = 500$. We notice that further increasing the two numbers brings no significant performance improvement. We leave the theoretical study of the impact of $k$ and $l$ to our future work. Another major computational cost comes from updating the weights of the inter-modality point pairs. The time complexity is $O(dN)$, where $N$ is the number of inter-modality point pairs.

To summarize, our algorithm scales linearly with the number of inter-modality point pairs and the data dimensionality. In practice, the number of inter-modality point pairs is usually small, making our algorithm very efficient.

## 2.4 Extensions

We briefly discuss two possible extensions of CRH in this subsection. First, we note that it is easy to extend CRH to learn nonlinear hash functions via the kernel trick [26]. Specifically, according to the generalized representer theorem [27], we can represent the projection vectors $\mathbf{w}_x$ and $\mathbf{w}_y$ as

$$\mathbf{w}_x = \sum_{i=1}^{I} \alpha_i \phi_x(\mathbf{x}_i) \text{ and } \mathbf{w}_y = \sum_{j=1}^{J} \beta_j \phi_y(\mathbf{y}_j),$$

where $\phi_x(\cdot)$ and $\phi_y(\cdot)$ are kernel-induced feature maps for modalities $\mathcal{X}$ and $\mathcal{Y}$, respectively. Then the objective function (1) can be expressed in kernel form and kernel-based hash functions can be learned by minimizing a new but very similar objective function.

Another possible extension is to make CRH support more than two modalities. Taking a new modality $\mathcal{Z}$ for example, we need to incorporate into Equation (1) the following terms: loss and regularization terms for $\mathcal{Z}$, and all pairwise loss terms involving $\mathcal{Z}$ and other modalities, e.g., $\mathcal{X}$ and $\mathcal{Y}$.

For both extensions, it is straightforward to adapt the algorithm presented above to solve the new optimization problems.

## 2.5 Discussions

CRH is closely related to a recent multimodal metric learning method called Multiview Neighborhood Preserving Projections (Multi-NPP) [23], because CRH uses a loss function for inter-modality point pairs which is similar to Multi-NPP. However, CRH is a general framework and other loss functions for inter-modality point pairs can also be adopted. The two methods have at least three significant differences. First, our focus is on HFL while Multi-NPP is on metric learning through embedding. Second, in addition to the inter-modality loss term, the objective function in CRH includes two intra-modality loss terms for large margin HFL while Multi-NPP only has a loss term for the inter-modality point pairs. Third, CRH uses boosting to sequentially learn the hash functions but Multi-NPP does not take this aspect into consideration.

As discussed briefly in [23], one may first use Multi-NPP to map multimodal data into a common real space and then apply any unimodal HFL method for multimodal hashing. However, this naive two-stage approach has some limitations. First, both stages can introduce information loss which impairs the quality of the hash functions learned. Second, a two-stage approach generally needs more computational resources. These two limitations can be overcome by using a one-stage method such as CRH.

## 3 Experiments

### 3.1 Experimental Settings

In our experiments, we compare CRH with two state-of-the-art multimodal hashing methods, namely, Cross-Modal Similarity Sensitive Hashing (CMSSH) [20][2] and Cross-View Hashing (CVH) [21],[3] for two cross-modal retrieval tasks: (1) *image query vs. text database*; (2) *text query vs. image database*. The goal of each retrieval task is to find from the text (image) database the nearest neighbors for the image (text) query.

We use two benchmark data sets which are, to the best of our knowledge, the largest fully paired and labeled multimodal data sets. We further divide each data set into a database set and a query set. To train the models, we randomly select a group of documents from the database set to form the training set. Moreover, we randomly select 0.1% of the point pairs from the training set. For fair comparison, all models are trained on the same training set and the experiments are repeated with 5 independent training sets.

The mean average precision (mAP) is used as the performance measure. To compute the mAP, we first evaluate the average precision (AP) of a set of $R$ retrieved documents by AP $= \frac{1}{L}\sum_{r=1}^{R} P(r)\,\delta(r)$, where $L$ is the number of true neighbors in the retrieved set, $P(r)$ denotes the precision of the top $r$ retrieved documents, and $\delta(r) = 1$ if the $r$th retrieved document is a true neighbor and $\delta(r) = 0$ otherwise. The mAP is then computed by averaging the AP values over all the queries in the query set. The larger the mAP, the better the performance. In the experiments, we set $R = 50$. Besides, we also report the precision and recall within a fixed Hamming radius.

We use cross-validation to choose the parameters for CRH and find that the model performance is only mildly sensitive to the parameters. As a result, in all experiments, we set $\lambda_x = 0.01, \lambda_y = 0.01, \gamma = 1000, a = 3.7$, and $\lambda = 1/a$. Besides, unless specified otherwise, we fix the training set size to 2,000 and the code length $K$ to 24.

### 3.2 Results on Wiki

The Wiki data set, generated from Wikipedia featured articles, consists of 2,866 image-text pairs.[4] In each pair, the text is an article describing some events or people and the image is closely related to

the content of the article. The images are represented by 128-dimensional SIFT [28] feature vectors, while the text articles are represented by the probability distributions over 10 topics learned by a latent Dirichlet allocation (LDA) model [29]. Each pair is labeled with one of 10 semantic classes. We simply use these class labels to identify the neighbors. Moreover, we use 80% of the data as the database set and the remaining 20% to form the query set.

The mAP values of the three methods are reported in Table 1. We can see that CRH outperforms CVH and CMSSH under all settings and CVH performs slightly better than CMSSH. We note that CMSSH ignores the intra-modality relational information and CVH simply treats each bit independently. Hence the performance difference is expected.

Table 1: mAP comparison on Wiki

| Task | Method | Code Length | | |
|---|---|---|---|---|
| | | $K = 24$ | $K = 48$ | $K = 64$ |
| Image Query | CRH | **$0.2537 \pm 0.0206$** | **$0.2399 \pm 0.0185$** | **$0.2392 \pm 0.0131$** |
| vs. | CVH | $0.2043 \pm 0.0150$ | $0.1788 \pm 0.0149$ | $0.1732 \pm 0.0072$ |
| Text Database | CMSSH | $0.1965 \pm 0.0123$ | $0.1780 \pm 0.0080$ | $0.1624 \pm 0.0073$ |
| Text Query | CRH | **$0.2896 \pm 0.0214$** | **$0.2882 \pm 0.0261$** | **$0.2989 \pm 0.0293$** |
| vs. | CVH | $0.2714 \pm 0.0164$ | $0.2304 \pm 0.0104$ | $0.2156 \pm 0.0202$ |
| Image Database | CMSSH | $0.2179 \pm 0.0161$ | $0.2094 \pm 0.0072$ | $0.2040 \pm 0.0135$ |

We further compare the three methods on several aspects in Figure 1. We first vary the size of the training set in subfigures 1(a) and 1(d). Although CVH performs the best when the training set is small, its performance is gradually surpassed by CRH as the size increases. We then plot the precision-recall curves and recall curves for all three methods in the remaining subfigures. It is clear that CRH outperforms its two counterparts by a large margin.

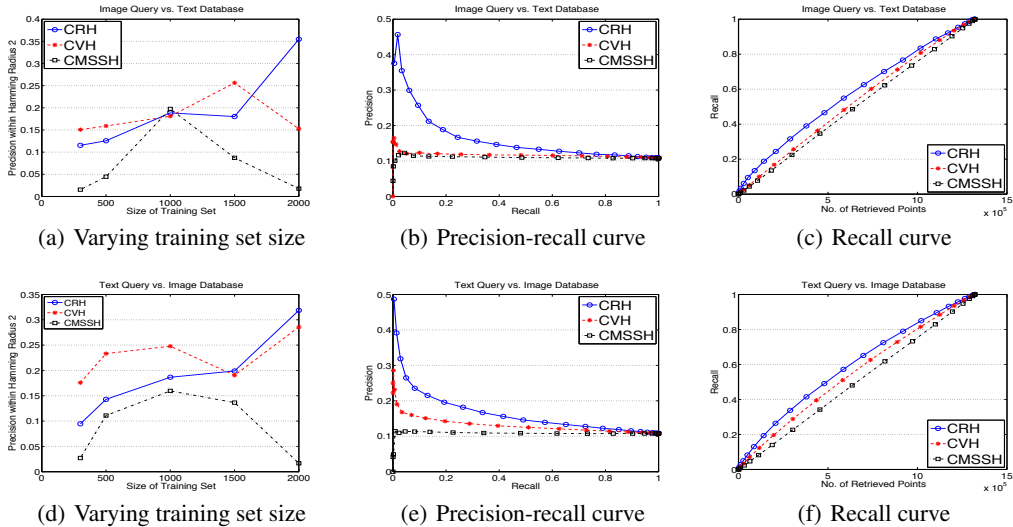

(a) Varying training set size     (b) Precision-recall curve     (c) Recall curve

(d) Varying training set size     (e) Precision-recall curve     (f) Recall curve

Figure 1: Results on Wiki

### 3.3 Results on Flickr

The Flickr data set consists of $186,577$ image-tag pairs pruned from the NUS data set[5] [30] by keeping the pairs that belong to one of the 10 largest classes. The images are represented by 500-dimensional SIFT vectors. To obtain more compact representations of the tags, we perform PCA on the original tag occurrence features and obtain 1000-dimensional feature vectors. Each pair is annotated by at least one of 10 semantic labels, and two points are defined as neighbors if they share at least one label. We use 99% of the data as the database set and the remaining 1% to form the query set.

The mAP values of the three methods are reported in Table 2. In the task of image query vs. text database, CRH performs comparably to CMSSH, which is better than CVH. However, in the other task, CRH achieves the best performance.

Table 2: mAP comparison on Flickr

| Task | Method | Code Length | | |
|---|---|---|---|---|
| | | $K = 24$ | $K = 48$ | $K = 64$ |
| Image Query vs. Text Database | CRH | $0.5259 \pm 0.0094$ | $0.4990 \pm 0.0075$ | $\mathbf{0.4929 \pm 0.0064}$ |
| | CVH | $0.4717 \pm 0.0035$ | $0.4515 \pm 0.0041$ | $0.4471 \pm 0.0023$ |
| | CMSSH | $\mathbf{0.5287 \pm 0.0123}$ | $\mathbf{0.5098 \pm 0.0141}$ | $0.4911 \pm 0.0220$ |
| Text Query vs. Image Database | CRH | $\mathbf{0.5364 \pm 0.0021}$ | $\mathbf{0.5185 \pm 0.0050}$ | $\mathbf{0.5064 \pm 0.0055}$ |
| | CVH | $0.4598 \pm 0.0020$ | $0.4519 \pm 0.0029$ | $0.4477 \pm 0.0058$ |
| | CMSSH | $0.5029 \pm 0.0321$ | $0.4815 \pm 0.0101$ | $0.4660 \pm 0.0298$ |

Similar to the previous subsection, we have conducted a group of experiments to compare the three methods on several aspects and report the results in Figure 2. The results for varying the size of the training set are plotted in subfigures 2(a) and 2(d). As more training data are used, CRH always performs better but the performance of CVH and CMSSH has high variance. The precision-recall curves and recall curves are shown in the remaining subfigures. Similar to the results on Wiki, CRH performs the best. However, the performance gap is smaller here.

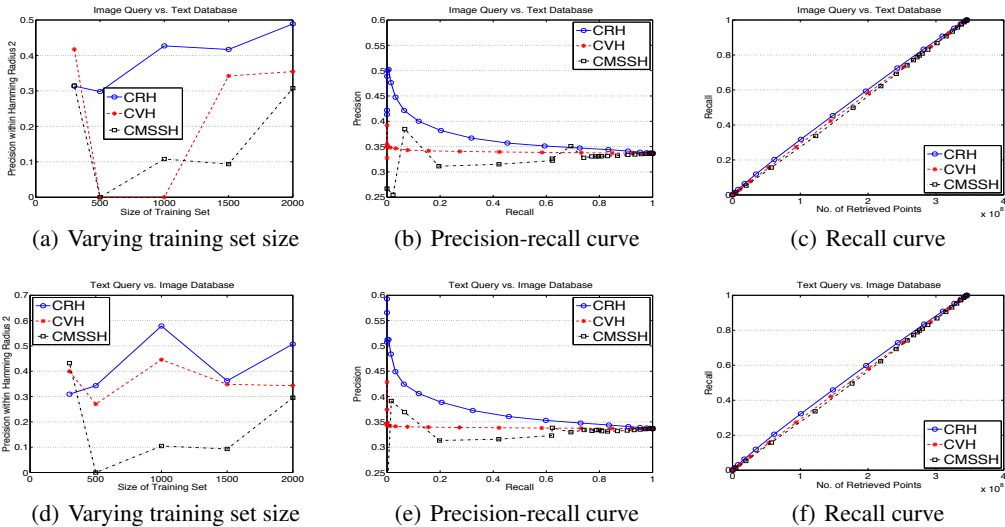

(a) Varying training set size     (b) Precision-recall curve     (c) Recall curve

(d) Varying training set size     (e) Precision-recall curve     (f) Recall curve

Figure 2: Results on Flickr

## 4 Conclusions

In this paper, we have presented a novel method for multimodal hash function learning based on a boosted co-regularization framework. Because the objective function of the optimization problem is in the form of a difference of convex functions, we can devise an efficient learning algorithm based on CCCP and a stochastic sub-gradient method. Comparative studies based on two benchmark data sets show that CRH outperforms two state-of-the-art multimodal hashing methods.

To take this work further, we would like to conduct theoretical analysis of CRH and apply it to some other tasks such as multimodal medical image alignment. Another possible research issue is to develop more efficient optimization algorithms to further improve the scalability of CRH.

## Acknowledgement

This research has been supported by General Research Fund 621310 from the Research Grants Council of Hong Kong.

## Footnotes

[1] For simplicity of our presentation, we focus on the bimodal case here and leave the discussion on extension to more than two modalities to Section 2.4.

[2]We used the implementation generously provided by the authors.

[3]We implemented the method ourselves because the code is not publicly available.

[4]http://www.svcl.ucsd.edu/projects/crossmodal/

[5]http://lms.comp.nus.edu.sg/research/NUS-WIDE.htm

# References

[1] Gregory Shakhnarovich, Trevor Darrell, and Piotr Indyk, editors. *Nearest-Neighbor Methods in Learning and Vision: Theory and Practice*. MIT Press, March 2006.

[2] Piotr Indyk and Rajeev Motwani. Approximate nearest neighbors: Towards removing the curse of dimensionality. In *STOC*, 1998.

[3] Moses Charikar. Similarity estimation techniques from rounding algorithms. In *STOC*, 2002.

[4] Alexandr Andoni and Piotr Indyk. Near-optimal hashing algorithms for approximate nearest neighbor in high dimensions. *Communications of the ACM*, 51(1):117–122, 2008.

[5] Brian Kulis and Kristen Grauman. Kernelized locality-sensitive hashing for scalable image search. In *ICCV*, 2009.

[6] Gregory Shakhnarovich, Paul Viola, and Trevor Darrell. Fast pose estimation with parameter-sensitive hashing. In *ICCV*, 2003.

[7] Ruslan Salakhutdinov and Geoffrey E. Hinton. Semantic hashing. In *SIGIR Workshop on Information Retrieval and Applications of Graphical Models*, 2007.

[8] Antonio Torralba, Rob Fergus, and Yair Weiss. Small codes and large image databases for recognition. In *CVPR*, 2008.

[9] Yair Weiss, Antonio Torralba, and Rob Fergus. Spectral hashing. In *NIPS 21*, 2008.

[10] Brian Kulis and Trevor Darrell. Learning to hash with binary reconstructive embeddings. In *NIPS 22*, 2009.

[11] Maxim Raginsky and Svetlana Lazebnik. Locality-sensitive binary codes from shift-invariant kernels. In *NIPS 22*, 2009.

[12] Ruei-Sung Lin, David A. Ross, and Jay Yagnik. SPEC hashing: Similarity preserving algorithm for entropy-based coding. In *CVPR*, 2010.

[13] Junfeng He, Wei Liu, and Shih-Fu Chang. Scalable similarity search with optimized kernel hashing. In *KDD*, 2010.

[14] Mohammad Norouzi and David J. Fleet. Minimal loss hashing for compact binary codes. In *ICML*, 2011.

[15] Jun Wang, Sanjiv Kumar, and Shih-Fu Chang. Semi-supervised hashing for scalable image retrieval. In *CVPR*, 2010.

[16] Yadong Mu, Jialie Shen, and Shuicheng Yan. Weakly-supervised hashing in kernel space. In *CVPR*, 2010.

[17] Dan Zhang, Fei Wang, and Luo Si. Composite hashing with multiple information sources. In *SIGIR*, 2011.

[18] Jingkuan Song, Yi Yang, Zi Huang, Heng Tao Shen, and Richang Hong. Multiple feature hashing for real-time large scale near-duplicate video retrieval. In *ACM MM*, 2011.

[19] Wei Liu, Jun Wang, Sanjiv Kumar, and Shih-Fu Chang. Hashing with graphs. In *ICML*, 2011.

[20] Michael M. Bronstein, Alexander M. Bronstein, Fabrice Michel, and Nikos Paragios. Data fusion through cross-modality metric learning using similarity-sensitive hashing. In *CVPR*, 2010.

[21] Shaishav Kumar and Raghavendra Udupa. Learning hash functions for cross-view similarity search. In *IJCAI*, 2011.

[22] A. L. Yuille and Anand Rangarajan. The concave-convex procedure (CCCP). In *NIPS 14*, 2001.

[23] Novi Quadrianto and Christoph H. Lampert. Learning multi-view neighborhood preserving projections. In *ICML*, 2011.

[24] Shai Shalev-Shwartz, Yoram Singer, and Nathan Srebro. Pegasos: Primal estimated sub-gradient solver for SVM. In *ICML*, 2007.

[25] Yoav Freund and Robert E. Schapire. A decision-theoretic generalization of on-line learning and an application to boosting. *Journal of Computer and System Sciences*, 55(1):119–139, 1997.

[26] John Shawe-Taylor and Nello Cristianini. *Kernel Methods for Pattern Analysis*. Cambridge University Press, 2004.

[27] Bernhard Schölkopf, Ralf Herbrich, and Alex J. Smola. A generalized representer theorem. In *COLT*, 2001.

[28] David G. Lowe. Distinctive image features from scale-invariant keypoints. *International Journal of Computer Vision*, 60(2):91–110, 2004.

[29] David M. Blei, Andrew Y. Ng, and Michael I. Jordan. Latent Dirichlet allocation. *Journal of Machine Learning Research*, 3:993–1022, 2003.

[30] Tat-Seng Chua, Jinhui Tang, Richang Hong, Haojie Li, Zhiping Luo, and Yan-Tao Zheng. NUS-WIDE: A real-world web image database from National University of Singapore. In *CIVR*, 2009.

